# A variational principle for model-based morphing

**Lawrence K. Saul\* and Michael I. Jordan**
Center for Biological and Computational Learning
Massachusetts Institute of Technology
79 Amherst Street, E10-034D
Cambridge, MA 02139

## Abstract

Given a multidimensional data set and a model of its density,
we consider how to define the optimal interpolation between two
points. This is done by assigning a cost to each path through space,
based on two competing goals—one to interpolate through regions
of high density, the other to minimize arc length. From this path
functional, we derive the Euler-Lagrange equations for extremal
motion; given two points, the desired interpolation is found by solv-
ing a boundary value problem. We show that this interpolation can
be done efficiently, in high dimensions, for Gaussian, Dirichlet, and
mixture models.

## 1 Introduction

The problem of non-linear interpolation arises frequently in image, speech, and
signal processing. Consider the following two examples: (i) given two profiles of the
same face, connect them by a smooth animation of intermediate poses[1]; (ii) given a
telephone signal masked by intermittent noise, fill in the missing speech. Both these
examples may be viewed as instances of the same abstract problem. In qualitative
terms, we can state the problem as follows[2]: given a multidimensional data set,
and two points from this set, find a smooth adjoining path that is consistent with
available models of the data. We will refer to this as the problem of *model-based
morphing*.

In this paper, we examine this problem it arises from statistical models of multi-
dimensional data. Specifically, our focus is on models that have been derived from

some form of density estimation. Though there exists a large body of work on the use of statistical models for regression and classification, there has been comparatively little work on the other types of operations that these models support. Non-linear morphing is an example of such an operation, one that has important applications to video email[3], low-bandwidth teleconferencing[4], and audiovisual speech recognition[2].

A common way to describe multidimensional data is some form of mixture modeling. Mixture models represent the data as a collection of two or more clusters; thus, they are well-suited to handling complicated (multimodal) data sets. Roughly speaking, for these models the problem of interpolation can be divided into two tasks—how to interpolate between points in the same cluster, and how to interpolate between points in different clusters. Our paper will therefore be organized along these lines.

Previous studies of morphing have exploited the properties of radial basis function networks[1] and locally linear models[2]. We have been influenced by both these works, especially in the abstract formulation of the problem. New features of our approach include: the fundamental role played by the density, the treatment of non-Gaussian models, the use of a continuous variational principle, and the description of the interpolant by a differential equation.

## 2   Intracluster interpolation

Let $Q = \{\mathbf{q}^{(1)}, \mathbf{q}^{(2)}, \ldots, \mathbf{q}^{|Q|}\}$ denote a set of multidimensional data points, and let $P(\mathbf{q})$ denote a model of the distribution from which these points were generated. Given two points, our problem is to find a smooth adjoining path that respects the statistical model of the data. In particular, the desired interpolant should not pass through regions of space that the modeled density $P(\mathbf{q})$ assigns low probability.

### 2.1   Clusters and metrics

To develop these ideas further, we begin by considering a special class of models— namely, those that represent clusters. We say that $P(\mathbf{q})$ models a *data cluster* if $P(\mathbf{q})$ has a unique (global) maximum; in turn, we identify the location of this maximum, $\mathbf{q}^*$, as the *prototype*.

Let us now consider the geometry of the space inhabited by the data. To endow this space with a geometric structure, we must define a metric, $g_{\alpha\beta}(\mathbf{q})$, that provides a measure of the distance between two nearby points:

$$\mathcal{D}[\mathbf{q}, \mathbf{q} + \mathbf{dq}] = \left[ \sum_{\alpha\beta} g_{\alpha\beta}(\mathbf{q}) \, dq_\alpha dq_\beta \right]^{\frac{1}{2}} + \mathrm{O}\left(|\mathbf{dq}|^2\right). \qquad (1)$$

Intuitively speaking, the metric should reflect the fact that as one moves away from the center of the cluster, the density of the data dies off more quickly in some directions than in others. A natural choice for the metric, one that meets the above criteria, is the negative Hessian of the log-likelihood:

$$g_{\alpha\beta}(\mathbf{q}) = -\frac{\partial^2}{\partial q_\alpha \partial q_\beta} \left[\ln P(\mathbf{q})\right]. \qquad (2)$$

This metric is positive-definite if $\ln P(\mathbf{q})$ is concave; this will be true for all the examples we discuss.

## 2.2 From densities to paths

The problem of model-based interpolation is to balance two competing goals—one to interpolate through regions of high density, the other to avoid excessive deformations. Using the metric in eq. (1), we can now assign a cost (or penalty) to each path based on these competing goals.

Consider the path parameterized by $\mathbf{q}(t)$. We begin by dividing the path into segments, each of which is traversed in some small time interval, $dt$. We assign a value to each segment by

$$\phi(t) = \left\{ \left[ \frac{P(\mathbf{q}(t))}{P(\mathbf{q}^*)} \right] e^{-\ell} \right\}^{\mathcal{D}[\mathbf{q}(t), \mathbf{q}(t+dt)]}, \tag{3}$$

where $\ell \geq 0$. For reasons that will become clear shortly, we refer to $\ell$ as the *line tension*. The value assigned to each segment depends on two terms: a ratio of probabilities, $P(\mathbf{q}(t))/P(\mathbf{q}^*)$, which favors points near the prototype, and the constant multiplier, $e^{-\ell}$. Both these terms are upper bounded by unity, and hence so is their product. The value of the segment also decays with its length, as a result of the exponent, $\mathcal{D}[\mathbf{q}(t), \mathbf{q}(t+dt)]$.

We derive a path functional by piecing these segments together, multiplying their individual contributions, and taking the continuum limit. A value for the entire path is obtained from the product:

$$e^{-\mathcal{S}} = \prod_t \phi(t). \tag{4}$$

Taking the logarithm of both sides, and considering the limit $dt \rightarrow 0$, we obtain the path functional

$$\mathcal{S}[\mathbf{q}(t)] = \int \left\{ -\ln \left[ \frac{P(\mathbf{q}(t))}{P(\mathbf{q}^*)} \right] + \ell \right\} \left[ \sum_{\alpha\beta} g_{\alpha\beta}(\mathbf{q}) \, \dot{q}_\alpha \dot{q}_\beta \right]^{\frac{1}{2}} dt, \tag{5}$$

where $\dot{\mathbf{q}} \equiv \frac{d}{dt}[\mathbf{q}]$ is the tangent vector to the path at time $t$. The terms in this functional balance the two competing goals for non-linear interpolation. The first favors paths that interpolate through regions of high density, while the second favors paths with small arc lengths; both are computed under the metric induced by the modeled density. The line tension $\ell$ determines the cost per unit arc length and modulates the competition between the two terms. Note that the value of the functional does not depend on the rate at which the path is traversed.

To minimize this functional, we use the following result from the calculus of variations. Let $\mathcal{L}(\mathbf{q}, \dot{\mathbf{q}})$ denote the integrand of eq. (5), such that $\mathcal{S}[\mathbf{q}(t)] = \int dt \, \mathcal{L}(\mathbf{q}, \dot{\mathbf{q}})$. Then the path which minimizes this functional obeys the Euler-Lagrange equations[5]:

$$\frac{d}{dt} \left( \frac{\partial \mathcal{L}}{\partial \dot{\mathbf{q}}} \right) = \frac{\partial \mathcal{L}}{\partial \mathbf{q}}. \tag{6}$$

We define the model-based interpolant between two points as the path which minimizes this functional; it is found by solving the associated boundary value problem. The function $\mathcal{L}(\mathbf{q}, \dot{\mathbf{q}})$ is known as the *Lagrangian*. In the next sections, we present eq. (5) for two distributions of interest—the multivariate Gaussian and the Dirichlet.

### 2.3   Gaussian cloud

The simplest model of multidimensional data is the multivariate Gaussian. In this case, the data is modeled by

$$P(\mathbf{x}) = \frac{|\mathbf{M}|^{1/2}}{(2\pi)^{N/2}} \exp\left\{ -\frac{1}{2} \left[ \mathbf{x}^T \mathbf{M} \mathbf{x} \right] \right\}, \tag{7}$$

where $\mathbf{M}$ is the inverse covariance matrix and $N$ is the dimensionality. Without loss of generality, we have chosen the coordinate system so that the mean of the data coincides with the origin. For the Gaussian, the mean also defines the location of the prototype; moreover, from eq. (2), the metric induced by this model is just the inverse covariance matrix. From eq. (5), we obtain the path functional:

$$\mathcal{S}[\mathbf{x}(t)] = \int \left\{ \frac{1}{2} \left[ \mathbf{x}^T \mathbf{M} \mathbf{x} \right] + \ell \right\} \left[ \dot{\mathbf{x}}^T \mathbf{M} \dot{\mathbf{x}} \right]^{\frac{1}{2}} dt, \tag{8}$$

To find a model-based interpolant, we seek the path that minimizes this functional. Because the functional is parameterization-invariant, it suffices to consider paths that are traversed at a constant (unit) rate: $\dot{\mathbf{x}}^T \mathbf{M} \dot{\mathbf{x}} = 1$. From eq. (6), we find that the optimal path with this parameterization satisfies:

$$\left\{ \frac{1}{2} \left[ \mathbf{x}^T \mathbf{M} \mathbf{x} \right] + \ell \right\} \ddot{\mathbf{x}} + \left[ \mathbf{x}^T \mathbf{M} \dot{\mathbf{x}} \right] \dot{\mathbf{x}} = \mathbf{x}. \tag{9}$$

This is a set of coupled non-linear equations for the components of $\mathbf{x}(t)$. However, note that at any moment in time, the acceleration, $\ddot{\mathbf{x}}$, can be expressed as a linear combination of the position, $\mathbf{x}$, and the velocity, $\dot{\mathbf{x}}$. It follows that the motion of $\mathbf{x}$ lies in a plane; in particular, it lies in the plane spanned by the initial conditions, $\mathbf{x}$ and $\dot{\mathbf{x}}$, at time $t = 0$. This enables one to solve the boundary value problem efficiently, even in very high dimensions.

Figure 1a shows some solutions to this boundary value problem for different values of the line tension, $\ell$. Note how the paths bend toward the origin, with the degree of curvature determined by the line tension, $\ell$.

### 2.4   Dirichlet simplex

For many types of data, the multivariate Gaussian distribution is not the most appropriate model. Suppose that the data points are vectors of positive numbers whose elements sum to one. In particular, we say that $\mathbf{w}$ is a *probability vector* if $\mathbf{w} = (w_1, w_2, \ldots, w_N) \in \mathcal{R}^N$, $w_\alpha > 0$ for all $\alpha$, and $\sum_\alpha w_\alpha = 1$. Clearly, the multivariate Gaussian is not suited to data of this form, since no matter what the mean and covariance matrix, it cannot assign zero probability to vectors outside the simplex. Instead, a more natural model is the Dirichlet distribution:

$$P(\mathbf{w}) = \Gamma(\theta) \prod_\alpha \frac{w_\alpha^{\theta_\alpha - 1}}{\Gamma(\theta_\alpha)}, \tag{10}$$

where $\theta_\alpha > 0$ for all $\alpha$, and $\theta \equiv \sum_\alpha \theta_\alpha$. Here, $\Gamma(\cdot)$ is the gamma function, and $\theta_\alpha$ are parameters that determine the statistics of $P(\mathbf{w})$. Note that $P(\mathbf{w}) = 0$ for vectors that are not probability vectors; in particular, the simplex constraints on $\mathbf{w}$ are implicit assumptions of the model.

We can rewrite the Dirichlet distribution in a more revealing form as follows. First, let $\mathbf{w}^*$ denote the probability vector with elements $w_\alpha^* = \theta_\alpha / \theta$. Then, making a change of variables from $\mathbf{w}$ to $\ln \mathbf{w}$, we have:

$$P(\ln \mathbf{w}) = \frac{1}{Z_\theta} \exp \left\{ - \theta \left[ \mathrm{KL} \left( \mathbf{w}^* \| \mathbf{w} \right) \right] \right\}, \tag{11}$$

where $Z_\theta$ is a normalization factor that depends on $\theta_\alpha$ (but not $\mathbf{w}$), and the quantity in the exponent is $\theta$ times the Kullback-Leibler (KL) divergence,

$$\mathrm{KL} \left( \mathbf{w}^* \| \mathbf{w} \right) = \sum_\alpha w_\alpha^* \ln \left[ \frac{w_\alpha^*}{w_\alpha} \right]. \tag{12}$$

The KL divergence measures the mismatch between $\mathbf{w}$ and $\mathbf{w}^*$, with $\mathrm{KL}(\mathbf{w}^* \| \mathbf{w}) = 0$ if and only if $\mathbf{w} = \mathbf{w}^*$. Since $\mathrm{KL}(\mathbf{w}^* \| \mathbf{w})$ has no other minima besides the one at $\mathbf{w}^*$, we shall say that $P(\ln \mathbf{w})$ models a data cluster *in the variable* $\ln \mathbf{w}$.

The metric induced by this modeled density is computed by following the prescription of eq. (2). For two nearby points inside the simplex, $\mathbf{w}$ and $\mathbf{w} + \mathbf{dw}$, the result of this prescription is that the squared distance is given by

$$ds^2 = \theta \sum_\alpha \frac{dw_\alpha^2}{w_\alpha}. \tag{13}$$

Up to a multiplicative factor of $2\theta$, eq. (13) measures the infinitesimal KL divergence between $\mathbf{w}$ and $\mathbf{w} + \mathbf{dw}$. This is a natural metric for vectors whose elements can be interpreted as probabilities.

The functional for non-linear interpolation is found by substituting the modeled density and the induced metric into eq. (5). For the Dirichlet distribution, this gives:

$$S[\mathbf{w}(t)] = \int \left\{ \theta \left[ \mathrm{KL} \left( \mathbf{w}^* \| \mathbf{w} \right) \right] + \ell \right\} \left[ \theta \sum_\alpha \frac{\dot{w}_\alpha^2}{w_\alpha} \right]^{\frac{1}{2}} dt. \tag{14}$$

Our problem is to find the path that minimizes this functional. Because the functional is parameterization-invariant, it again suffices to consider paths that are traversed at a constant rate, or $\sum_\alpha \dot{w}_\alpha^2 / w_\alpha = 1$. In addition to this, however, we must also enforce the constraint that $\mathbf{w}$ remains inside the simplex; this is done by introducing a Lagrange multiplier. Following this procedure, we find that the optimal path is described by:

$$\left[ \theta \, \mathrm{KL}(\mathbf{w}^* \| \mathbf{w}) + \ell \right] \left\{ \ddot{w}_\alpha - \frac{\dot{w}_\alpha^2}{2 w_\alpha} + \frac{w_\alpha}{2} \right\} - \theta \left[ \sum_\beta \frac{w_\beta^*}{w_\beta} \dot{w}_\beta \right] \dot{w}_\alpha = \theta (w_\alpha - w_\alpha^*). \tag{15}$$

Given two endpoints, this differential equation defines a boundary value problem for the optimal path. Unlike before, however, in this case the motion of $\mathbf{w}$ is not

confined to a plane. Hence, the boundary value problem for eq. (15) does not collapse to one dimension, as does its Gaussian counterpart, eq. (9).

To remedy this situation, we have developed an efficient approximation that finds a near-optimal interpolant, in lieu of the optimal one. This is done in two steps: first, by solving eq. (15) exactly in the limit $\ell \to \infty$; second, by using this limiting solution, $\mathbf{w}^\infty(t)$, to find the lowest-cost path that can be expressed as the convex combination:

$$\mathbf{w}(t) = m(t)\mathbf{w}^* + [1 - m(t)]\,\mathbf{w}^\infty(t). \tag{16}$$

The lowest-cost path of this form is found by substituting eq. (16) into the Dirichlet functional, eq. (14), and solving the Euler-Lagrange equations for $m(t)$. The motivation for eq. (16) is that for finite $\ell$, we expect the optimal interpolant to deviate from $\mathbf{w}^\infty(t)$ and bend toward the prototype at $\mathbf{w}^*$. In practice, this approximation works very well, and by collapsing the boundary value problem to one dimension, it allows cheap computation of the Dirichlet interpolants. Some paths from eq. (16), as well as the $\ell \to \infty$ paths on which they are based, are shown in figure 1b. These paths were computed for the twelve dimensional simplex ($N = 12$), then projected onto the $w_1 w_2$-plane.

## 3  Intercluster interpolation

The Gaussian and Dirichlet distributions of the previous section are clearly inadequate for modeling for multimodal data sets. In this section, we extend the variational principle to mixture models, which describe the data as a collection of $k \geq 2$ clusters. In particular, suppose the data is modeled by

$$P(\mathbf{q}) = \sum_{z=1}^{k} \pi_z P(\mathbf{q}|z). \tag{17}$$

Here, we have assumed that the conditional densities $P(\mathbf{q}|z)$ model data clusters as defined in section 2.1, and the coefficients $\pi_z = P(z)$ define prior probabilities for the latent variable, $z \in \{1, 2, \ldots, k\}$.

The crucial step for mixture models is to develop the appropriate generalization of eq. (5). To this end, let $\mathcal{L}_z(\mathbf{q}, \dot{\mathbf{q}})$ denote the Lagrangian derived from the conditional density, $P(\mathbf{q}|z)$, and $\ell_z$ the line tension[1] that appears in this Lagrangian. We now combine these Lagrangians into a single functional:

$$\mathcal{S}[\mathbf{q}(t), z(t)] = \int dt\, \mathcal{L}_{z(t)}(\mathbf{q}, \dot{\mathbf{q}}). \tag{18}$$

Note that eq. (18) is a functional of two arguments, not one. For mixture models, which define a joint density $P(\mathbf{q}, z) = \pi_z P(\mathbf{q}|z)$, our goal is to find the optimal path in the joint space $\mathbf{q} \otimes z$. Here, $z(t)$ is a piecewise-constant function of time that assigns a discrete label to each point along the path; in other words, it provides a temporal segmentation of the path, $q(t)$. The purpose of $z(t)$ in eq. (18) is to select which Lagrangian is used to compute the contribution from the interval $[t, t + dt]$.

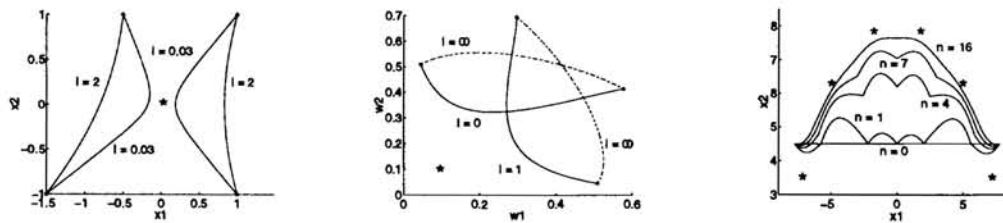

Figure 1: Model-based morphs for (a) Gaussian distribution; (b) Dirichlet distribution; (c) mixture of Gaussians. The prototypes are shown as asterisks; $\ell$ denotes the line tension. Figure 1c shows the convergence of the iterative algorithm; $n$ denotes the number of iterations.

As before, we define the model-based interpolant as the path $\mathbf{q}(t)$ that minimizes eq. (18). In this case, however, both $\mathbf{q}(t)$ and $z(t)$ must be simultaneously optimized to recover this path. We have implemented an iterative scheme to perform this optimization, one that alternately (i) estimates the segmentation $z(t)$, (ii) computes the model-based interpolant within each cluster based on this segmentation, and (iii) reestimates the points (along the cluster boundaries) where $z(t)$ changes value. In short, the strategy is to optimize $z(t)$ for fixed $\mathbf{q}(t)$, then optimize $\mathbf{q}(t)$ for fixed $z(t)$.

Figure 1c shows how this algorithm operates on a simple mixture of Gaussians. In this example, the covariance matrices were set equal to the identity matrix, and the means of the Gaussians were distributed along a circle in the $x_1 x_2$–plane. Note that with each iteration, the interpolant converges more closely to the path that traverses this circle. The effect is similar to the manifold-snake algorithm of Bregler and Omohundro[2].

## 4 Discussion

In this paper we have proposed a variational principle for model-based interpolation. Our framework handles Gaussian, Dirichlet, and mixture models, and the resulting algorithms scale well to high dimensions. Future work will concentrate on the application to real images.

## Footnotes

\*Current address: AT&T Labs, 600 Mountain Ave 2D–439, Murray Hill, NJ 07974

[1]To respect the weighting of the mixture components in eq. (17), we set the line tensions according to $\ell_z = \ell - \ln \pi_z$. Thus, components with higher weights have lower line tensions.

## References

[1] T. Poggio and F. Girosi. Networks for approximation and learning. *Proc. of IEEE*, vol 78:9 (1990).

[2] C. Bregler and S. Omohundro. Nonlinear image interpolation using manifold learning. In G. Tesauro, D. Touretzky, and T. Leen (eds.). *Advances in Neural Information Processing Systems 7*, 973–980. MIT Press, Cambridge, MA (1995).

[3] T. Ezzat. Example based analysis and synthesis for images of faces. *MIT EECS M.S. thesis* (1996).

[4] D. Beymer, A. Shashua, and T. Poggio. Example based image analysis and synthesis. *AI Memo 1161*, MIT (1993).

[5] H. Goldstein. *Classical Mechanics.* Addison-Wesley, London (1980).
